# Adaptive Martingale Boosting

**Philip M. Long**
Google
plong@google.com

**Rocco A. Servedio**
Columbia University
rocco@cs.columbia.edu

## Abstract

In recent work Long and Servedio [LS05] presented a "martingale boosting" algorithm that works by constructing a branching program over weak classifiers and has a simple analysis based on elementary properties of random walks. [LS05] showed that this martingale booster can tolerate random classification noise when it is run with a noise-tolerant weak learner; however, a drawback of the algorithm is that it is not *adaptive*, i.e. it cannot effectively take advantage of variation in the quality of the weak classifiers it receives.

We present an adaptive variant of the martingale boosting algorithm. This adaptiveness is achieved by modifying the original algorithm so that the random walks that arise in its analysis have different step size depending on the quality of the weak learner at each stage. The new algorithm inherits the desirable properties of the original [LS05] algorithm, such as random classification noise tolerance, and has other advantages besides adaptiveness: it requires polynomially fewer calls to the weak learner than the original algorithm, and it can be used with confidence-rated weak hypotheses that output real values rather than Boolean predictions.

## 1  Introduction

Boosting algorithms are efficient procedures that can be used to convert a weak learning algorithm (one which outputs a weak hypothesis that performs only slightly better than random guessing for a binary classification task) into a strong learning algorithm (one which outputs a high-accuracy classifier). A rich theory of boosting has been developed over the past two decades; see [Sch03, MR03] for some overviews. Two important issues for boosting algorithms which are relevant to the current work are *adaptiveness* and *noise-tolerance*; we briefly discuss each of these issues before describing the contributions of this paper.

**Adaptiveness.**  "Adaptiveness" refers to the ability of boosting algorithms to adjust to different accuracy levels in the sequence of weak hypotheses that they are given. The first generation of boosting algorithms [Sch90, Fre95] required the user to input an "advantage" parameter $\gamma$ such that the weak learner was guaranteed to always output a weak hypothesis with accuracy at least $1/2 + \gamma$. Given an initial setting of $\gamma$, even if the sequence of weak classifiers generated by the runs of the weak learner included some hypotheses with accuracy (perhaps significantly) better than $1/2+\gamma$, the early boosting algorithms were unable to capitalize on this extra accuracy; thus, these early boosters were not adaptive. Adaptiveness is an important property since it is often the case that the advantage of successive weak classifiers grows smaller and smaller as boosting proceeds.

A major step forward was the development of the AdaBoost algorithm [FS97]. AdaBoost does not require a lower bound $\gamma$ on the minimum advantage, and the error rate of its final hypothesis depends favorably on the different advantages of the different weak classifiers in the sequence. More precisely, if the accuracy of the $t$-th weak classifier is $\frac{1}{2} + \gamma_t$, then the AdaBoost final hypothesis

has error at most $\prod_{t=0}^{T-1} \sqrt{1 - 4\gamma_t^2}$. This error rate is usually upper bounded (see [FS97]) by

$$\exp\left(-2\sum_{t=0}^{T-1} \gamma_t^2\right) \tag{1}$$

and indeed (1) is a good approximation if no $\gamma_t$ is too large.

**Noise tolerance.** One drawback of many standard boosting techniques, including AdaBoost, is that they can perform poorly when run on noisy data [FS96, MO97, Die00, LS08]. Motivated in part by this observation, in recent years boosting algorithms that work by constructing *branching programs* over the weak classifiers (note that this is in contrast with AdaBoost, which constructs a single weighted sum of weak classifiers) have been developed and shown to enjoy some provable noise tolerance. In particular, the algorithms of [KS05, LS05] have been shown to boost to optimally high accuracy in the presence of random classification noise when run with a random classification noise tolerant weak learner. (Recall that "random classification noise at rate $\eta$" means that the true binary label of each example is independently flipped with probability $\eta$. This is a very well studied noise model, see e.g. [AL88, Kea98, AD98, BKW03, KS05, RDM06] and many other references.)

While the noise tolerance of the boosters [KS05, LS05] is an attractive feature, a drawback of these algorithms is that they do not enjoy the adaptiveness of algorithms like AdaBoost. The MMM booster of [KS05] is not known to have any adaptiveness at all, and the "martingale boosting" algorithm of [LS05] only has the following limited type of adaptiveness. The algorithm works in stages $t = 0, 1, \dots$ where in the $t$-th stage a collection of $t + 1$ weak hypotheses are obtained; let $\gamma_t$ denote the minimum advantage of these $t + 1$ hypotheses obtained in stage $t$. [LS05] shows that the final hypothesis constructed by martingale boosting has error at most

$$\exp\left(-\frac{(\sum_{t=0}^{T-1} \gamma_t)^2}{2T}\right). \tag{2}$$

(2) is easily seen to always be a worse bound than (1), and the difference can be substantial. Consider, for example, a sequence of weak classifiers in which the advantages decrease as $\gamma_t = 1/\sqrt{t+1}$ (this is in line with the oft-occurring situation, mentioned above, that advantages grow smaller and smaller as boosting progresses). For any $\epsilon > 0$ we can bound (1) from above by $\epsilon$ by taking $T \approx 1/\sqrt{\epsilon}$, whereas for this sequence of advantages the error bound (2) is never less than 0.5 (which is trivial), and in fact (2) approaches 1 as $t \to \infty$.

**Our contributions: adaptive noise-tolerant boosting.** We give the first boosting algorithm that is both adaptive enough to satisfy a bound of $\exp\left(-\Omega\left(\sum_{t=0}^{T-1} \gamma_t^2\right)\right)$ and is provably tolerant to random classification noise. We do this by modifying the martingale boosting algorithm of [LS05] to make it adaptive; the modification inherits the noise-tolerance of the original [LS05] algorithm. In addition to its adaptiveness, the new algorithm also improves on [LS05] by constructing a branching program with polynomially fewer nodes than the original martingale boosting algorithm (thus it requires fewer calls to the weak learner), and it can be used directly with weak learners that generate confidence-rated weak hypotheses (the original martingale boosting algorithm required the weak hypotheses to be Boolean-valued).

**Our approach.** We briefly sketch the new idea that lets us achieve adaptiveness. Recall that the original martingale booster of Long and Servedio formulates the boosting process as a random walk; intuitively, as a random example progresses down through the levels of the branching program constructed by the [LS05] booster, it can be viewed as performing a simple random walk with step size 1 on the real line, where the walk is biased in the direction (positive or negative) corresponding to the correct classification of the example. (The quantity tracked during the random walk is the difference between the number of positive predictions and the number of negative predictions made by base classifiers encountered in the braching program up to a given point in time.) This means that after enough stages, a random positive example will end up to the right of the origin with high probability, and contrariwise for a random negative example. Thus a high-accuracy classifier is obtained simply by labelling each example according to the sign ($+$ or $-$) of its final location on the real line.

The new algorithm extends this approach in a simple and intuitive way, by having examples perform a random walk *with variable step size*: if the weak classifier at a given internal node has large

advantage, then the new algorithm makes the examples that reach that node take a large step in the random walk. This is a natural way to exploit the fact that examples reaching such a large-advantage node usually tend to walk in the right direction. The idea extends straightforwardly to let us handle *confidence-rated* weak hypotheses (see [SS99]) whose predictions are real values in $[-1, 1]$ as opposed to Boolean values from $\{-1, 1\}$. This is done simply by scaling the step size for a given example $x$ from a given node according to the numerical value $h(x)$ that the confidence-rated weak hypothesis $h$ at that node assigns to example $x$.

While using different step sizes at different levels is a natural idea, it introduces some complications. In particular, if a branching program is constructed naively based on this approach, it is possible for the number of nodes to increase exponentially with the depth. To avoid this, we use a randomized rounding scheme together with the variable-step random walk to ensure that the number of nodes in the branching program grows polynomially rather than exponentially in the number of stages in the random walk (i.e. the depth of the branching program). In fact, we actually improve on the efficiency of the original martingale boosting algorithm of [LS05] by a polynomial factor, by truncating "extreme" nodes in the branching program that are "far" from the origin. Our analysis shows that this truncation has only a small effect on the accuracy of the final classifier, while giving a significant asymptotic savings in the size of the final branching program (roughly $1/\gamma^3$ nodes as opposed to the $1/\gamma^4$ nodes of [KS05, LS05]).

## 2   Preliminaries

We make the following assumptions and notational conventions throughout the paper. There is an initial distribution $\mathcal{D}$ over a domain of examples $X$. There is a target function $c : X \to \{-1, 1\}$ that we are trying to learn. Given the target function $c$ and the distribution $\mathcal{D}$, we write $\mathcal{D}^+$ to denote the distribution $\mathcal{D}$ restricted to the positive examples $\{x \in X : c(x) = 1\}$. Thus, for any event $S \subseteq \{x \in X : c(x) = 1\}$ we have $\mathbf{Pr}_{\mathcal{D}+}[x \in S] = \mathbf{Pr}_{\mathcal{D}}[x \in S]/\mathbf{Pr}_{\mathcal{D}}[c(x) = 1]$. Similarly, we write $\mathcal{D}^-$ to denote $\mathcal{D}$ restricted to the negative examples $\{x \in X : c(x) = -1\}$.

As usual, our boosting algorithms work by repeatedly passing a distribution $\mathcal{D}'$ derived from $\mathcal{D}$ to a weak learner, which outputs a classifier $h$. The future behavior will be affected by how well $h$ performs on data distributed according to $\mathcal{D}'$. To keep the analysis clean, we will abstract away issues of sampling from $\mathcal{D}'$ and estimating the accuracy of the resulting $h$. These issues are trivial if $\mathcal{D}$ is uniform over a moderate-sized domain (since all probabilities can be computed exactly), and otherwise they can be handled via the same standard estimation techniques used in [LS05].

**Martingale boosting.** We briefly recall some key aspects of the martingale boosting algorithm of [LS05] which are shared by our algorithm (and note some differences). Both boosters work by constructing a leveled branching program. Each node in the branching program has a *location*; this is a pair $(\beta, t)$ where $\beta$ is a real value (a location on the line) and $t \geq 0$ is an integer (the level of the node; each level corresponds to a distinct stage of boosting). The initial node, where all examples start, is at $(0, 0)$. In successive stages $t = 0, 1, 2, \dots$ the booster constructs nodes in the branching program at levels $0, 1, 2, \dots$. For a location $(\beta, t)$ where the branching program has a node, let $\mathcal{D}_{\beta, t}$ be the distribution $\mathcal{D}$ conditioned on reaching the node at $(\beta, t)$. We sometimes refer to this distribution $\mathcal{D}_{\beta, t}$ as the *distribution induced by node* $(\beta, t)$.

As boosting proceeds, in stage $t$, each node $(\beta, t)$ at level $t$ is assigned a hypothesis which we call $h_{\beta, t}$. Unlike [LS05] we shall allow confidence-rated hypotheses, so each weak hypothesis is a mapping from $X$ to $[-1, 1]$. Once the hypothesis $h_{\beta, t}$ has been obtained, out-edges are constructed from $(\beta, t)$ to its child nodes at level $t + 1$. While the original martingale boosting algorithm of [LS05] had two child nodes at $(\beta - 1, t + 1)$ and $(\beta + 1, t + 1)$ from each internal node, as we describe in Section 3 our new algorithm will typically have *four* child nodes for each node (but may, for a confidence-rated base classifier, have as many as eight).

**Our algorithm.** To fully specify our new boosting algorithm we must describe:

(1) How the weak learner is run at each node $(\beta, t)$ to obtain a weak classifier. This is straightforward for the basic case of "two-sided" weak learners that we describe in Section 3 and somewhat less straightforward in the usual (non-two-sided) weak learner setting. In Section 5.1 we describe how to use a standard weak learner, and how to handle noise – both extensions borrow heavily from earlier work [LS05, KS05].

(2) What function is used to label the node $(\beta, t)$, i.e. how to route subsequent examples that reach $(\beta, t)$ to one of the child nodes. It turns out that this function is a randomized version of the weak classifier mentioned in point (1) above.

(3) Where to place the child nodes at level $t + 1$; this is closely connected with (2) above.

As in [LS05], once the branching program has been fully constructed down through some level $T$ the final hypothesis it computes is very simple. Given an input example $x$, the output of the final hypothesis on $x$ is $\text{sgn}(\beta)$ where $(\beta, T)$ is the location in level $T$ to which $x$ is ultimately routed as it passes through the branching program.

## 3 Boosting a two-sided weak learner

In this section we assume that we have a *two-sided weak learner*. This is an algorithm which, given a distribution $\mathcal{D}$, can always obtain hypotheses that have *two-sided advantage* as defined below:

**Definition 1** *A hypothesis* $h : X \to [-1, 1]$ *has* two-sided advantage $\gamma$ *with respect to* $\mathcal{D}$ *if it satisfies both* $\mathbf{E}_{x \in \mathcal{D}^+}[h(x)] \geq \gamma$ *and* $\mathbf{E}_{x \in \mathcal{D}^-}[h(x)] \leq -\gamma$.

As we explain in Section 5.1 we may apply methods of [LS05] to reduce the typical case, in which we only receive "normal" weak hypotheses rather than two-sided weak hypotheses, to this case.

The branching program starts off with a single node at location $(0, 0)$. Assuming the branching program has been constructed up through level $t$, we now explain how it is extended in the $t$-th stage up through level $t + 1$. There are two basic steps in each stage: weak training and branching.

**Weak training.** Consider a given node at location $(\beta, t)$ in the branching program. As in [LS05] we construct a weak hypothesis $h_{\beta,t}$ simply by running the two-sided weak learner on examples drawn from $\mathcal{D}_{\beta,t}$ and letting $h_{\beta,t}$ be the hypothesis it generates. Let us write $\gamma_{\beta,t}$ to denote

$$\gamma_{\beta,t} \overset{\text{def}}{=} \min\{\mathbf{E}_{x \in (\mathcal{D}_{\beta,t})^+}[h_{\beta,t}(x)], \mathbf{E}_{x \in (\mathcal{D}_{\beta,t})^-}[-h_{\beta,t}(x)]\}.$$

We call $\gamma_{\beta,t}$ the *advantage* at node $(\beta, t)$.

We do this for all nodes at level $t$. Now we define the *advantage at level $t$* to be

$$\gamma_t \overset{\text{def}}{=} \min_{\beta} \gamma_{\beta,t}. \tag{3}$$

**Branching.** Intuitively, we would like to use $\gamma_t$ as a scaling factor for the "step size" of the random walk at level $t$. Since we are using confidence-rated weak hypotheses, it is also natural to have the step that example $x$ takes at a given node be proportional to the value of the confidence-rated hypothesis at that node on $x$. The most direct way to do this would be to label the node $(\beta, t)$ with the weak classifier $h_{\beta,t}$ and to route each example $x$ to a node at location $(\beta + \gamma_t h_{\beta,t}(x), t + 1)$. However, there are obvious difficulties with this approach; for one thing a single node at $(\beta, t)$ could give rise to arbitrarily many (infinitely many, if $|X| = \infty$) nodes at level $t+1$. Even if the hypotheses $h_{\beta,t}$ were all guaranteed to $\{-1, 1\}$-valued, if we were to construct a branching program in this way then it could be the case that by the $T$-th stage there are $2^{T-1}$ distinct nodes at level $T$.

We get around this problem by creating nodes at level $t+1$ only at integer multiples of $\frac{\gamma_t}{2}$. Note that this "granularity" that is used is different at each level, depending on the advantage at each level (we shall see in the next section that this is crucial for the analysis). This keeps us from having too many nodes in the branching program at level $t + 1$. Of course, we only actually create those nodes in the branching program that have an incoming edge as described below (later we will give an analysis to bound the number of such nodes).

We simulate the effect of having an edge from $(\beta, t)$ to $(\beta + \gamma_t h_{\beta,t}(x), t + 1)$ by using *two* edges from $(\beta, t)$ to $(i \cdot \gamma_t/2, t + 1)$ and to $((i + 1) \cdot \gamma_t/2, t + 1)$, where $i$ is the unique integer such that $i \cdot \gamma_t/2 \leq \beta + \gamma_t h_{\beta,t}(x) < (i+1) \cdot \gamma_t/2$. To simulate routing an example $x$ to $(\beta + \gamma_t h_{\beta,t}(x), t+1)$, the branching program routes $x$ randomly along one of these two edges so that the expected location at which $x$ ends up is $(\beta + \gamma_t h_{\beta,t}(x), t + 1)$. More precisely, if $\beta + \gamma_t h_{\beta,t}(x) = (i + \rho) \cdot \gamma_t/2$ where $0 \leq \rho < 1$, then the rule used at node $(\beta, t)$ to route an example $x$ is "with probability $\rho$ send $x$ to $((i + 1) \cdot \gamma_t/2, t + 1)$ and with probability $(1 - \rho)$ send $x$ to $(i \cdot \gamma_t/2, t + 1)$."

Since $|h_{\beta,t}(x)| \leq 1$ for all $x$ by assumption, it is easy to see that at most eight outgoing edges are required from each node $(\beta, t)$. Thus the branching program that the booster constructs uses a randomized variant of each weak hypothesis $h_{\beta,t}$ to route examples along one of (at most) eight outgoing edges.

## 4 Proof of correctness for boosting a two-sided weak learner

The following theorem shows that the algorithm described above is an effective adaptive booster for two-sided weak learners:

**Theorem 2** *Consider running the above booster for $T$ stages. For $t = 0, \ldots, T - 1$ let the values $\gamma_0, \ldots, \gamma_{T-1} > 0$ be defined as described above, so each invocation of the two-sided weak learner on distribution $\mathcal{D}_{\beta,t}$ yields a hypothesis $h_{\beta,t}$ that has $\gamma_{\beta,t} \geq \gamma_t$. Then the final hypothesis $h$ constructed by the booster satisfies*

$$\mathbf{Pr}_{x \in \mathcal{D}}[h(x) \neq c(x)] \leq \exp\left(-\frac{1}{8}\sum_{t=0}^{T-1}\gamma_t^2\right). \tag{4}$$

*The algorithm makes at most $M \leq O(1) \cdot \sum_{t=0}^{T-1} \frac{1}{\gamma_t} \sum_{j=0}^{t-1} \gamma_j$ calls to the weak learner (i.e. constructs a branching program with at most $M$ nodes).*

**Proof:** We will show that $\mathbf{Pr}_{x \in \mathcal{D}^+}[h(x) \neq 1] \leq \exp\left(-\frac{1}{8}\sum_{t=0}^{T-1}\gamma_t^2\right)$; a completely symmetric argument shows a similar bound for negative examples, which gives (4).

For $t = 1, \ldots, T$ we define the random variable $A_t$ as follows: given a draw of $x$ from $\mathcal{D}^+$ (the original distribution $\mathcal{D}$ restricted to positive examples), the value of $A_t$ is $\gamma_{t-1}h_{\beta,t-1}(x)$, where $(\beta, t-1)$ is the location of the node that $x$ reaches at level $t$ of the branching program. Intuitively $A_t$ captures the direction and size of the move that we would like $x$ to make during the branching step that brings it to level $t$.

We define $B_t$ to be the random variable that captures the direction and size of the move that $x$ *actually* makes during the branching step that brings it to level $t$. More precisely, let $i$ be the integer such that $i \cdot (\gamma_{t-1}/2) \leq \beta + \gamma_{t-1}h_{\beta,t-1}(x) < (i+1) \cdot (\gamma_{t-1}/2)$, and let $\rho \in [0,1)$ be such that $\beta + \gamma_{t-1}h_{\beta,t-1}(x) = (i+\rho) \cdot (\gamma_{t-1}/2)$. Then

$$B_t = \begin{cases} ((i+1) \cdot (\gamma_{t-1}/2) - \beta) & \text{with probability } \rho, \text{ and} \\ (i \cdot (\gamma_{t-1}/2) - \beta) & \text{with probability } 1 - \rho. \end{cases}$$

We have that $\mathbf{E}[B_t]$ (where the expectation is taken only over the $\rho$-probability in the definition of $B_t$) equals $((i+\rho) \cdot (\gamma_{t-1}/2) - \beta)h_{\beta,t-1}(x) = \gamma_{t-1}h_{\beta,t-1}(x) = A_t$. Let $X_t$ denote $\sum_{i=1}^{t} B_i$, so the value of $X_t$ is the actual location on the real line where $x$ ends up at level $t$.

Fix $1 \leq t \leq T$ and let us consider the conditional random variable $(X_t|X_{t-1})$. Conditioned on $X_{t-1}$ taking any particular value (i.e. on $x$ reaching any particular location $(\beta, t-1)$), we have that $x$ is distributed according to $(\mathcal{D}_{\beta,t-1})^+$, and thus we have

$$\mathbf{E}[X_t|X_{t-1}] = X_{t-1} + \mathbf{E}_{x \in (\mathcal{D}_{\beta,t})^+}[\gamma_{t-1}h_{\beta,t-1}(x)] \geq X_{t-1} + \gamma_{t-1}\gamma_{\beta,t-1} \geq X_{t-1} + \gamma_{t-1}^2, \tag{5}$$

where the first inequality follows from the two-sided advantage of $h_{\beta,t-1}$.

For $t = 0, \ldots, T$, define the random variable $Y_t$ as $Y_t = X_t - \sum_{i=0}^{t-1} \gamma_i^2$ (so $Y_0 = X_0 = 0$). Since conditioning on the value of $Y_{t-1}$ is equivalent to conditioning on the value of $X_{t-1}$, using (5) we get

$$\mathbf{E}[Y_t|Y_{t-1}] = \mathbf{E}\left[X_t - \sum_{i=0}^{t-1}\gamma_i^2 \Big| Y_{t-1}\right] = \mathbf{E}[X_t|Y_{t-1}] - \sum_{i=0}^{t-1}\gamma_i^2 \geq X_{t-1} - \sum_{i=0}^{t-2}\gamma_i^2 = Y_{t-1},$$

so the sequence of random variables $Y_0, \ldots, Y_T$ is a sub-martingale.[1] To see that this sub-martingale has bounded differences, note that we have

$$|Y_t - Y_{t-1}| = |X_t - X_{t-1} - \gamma_{t-1}^2| = |B_t - \gamma_{t-1}^2|.$$

The value of $B_t$ is obtained by first moving by $\gamma_{t-1}h_{\beta,t-1}(x)$, and then rounding to a neighboring multiple of $\gamma_{t-1}/2$, so $|B_t| \leq (3/2)\gamma_{t-1}$, which implies $|Y_t - Y_{t-1}| \leq (3/2)\gamma_{t-1} + \gamma_{t-1}^2 \leq 2\gamma_{t-1}$.

Now recall Azuma's inequality for sub-martingales:

> *Let $0 = Y_0, \ldots, Y_T$ be a sub-martingale which has $|Y_i - Y_{i-1}| \leq c_i$ for each $i = 1, \ldots, T$. Then for any $\lambda > 0$ we have $\mathbf{Pr}[Y_T \leq -\lambda] \leq \exp\left(-\frac{\lambda^2}{2\sum_{i=1}^{T} c_i^2}\right)$.*

We apply this with each $c_i = 2\gamma_{i-1}$ and $\lambda = \sum_{t=0}^{T-1}\gamma_t^2$. This gives us that the error rate of $h$ on positive examples, $\mathbf{Pr}_{x\in\mathcal{D}^+}[h(x) = -1]$, equals

$$\mathbf{Pr}[X_T < 0] = \mathbf{Pr}[Y_T < -\lambda] \leq \exp\left(-\frac{\lambda^2}{8\sum_{t=0}^{T-1}\gamma_t^2}\right) = \exp\left(-\frac{1}{8}\sum_{t=0}^{T-1}\gamma_t^2\right). \tag{6}$$

So we have established (4); it remains to bound the number of nodes constructed in the branching program. Let us write $M_t$ to denote the number of nodes at level $t$, so $M = \sum_{t=0}^{T-1} M_t$.

The $t$-th level of boosting can cause the rightmost (leftmost) node to be at most $2\gamma_{t-1}$ distance farther away from the origin than the rightmost (leftmost) node at the $(t-1)$-st level. This means that at level $t$, every node is at a position $(\beta, t)$ with $|\beta| \leq 2\sum_{j=0}^{t-1}\gamma_j$. Since nodes are placed at integer multiples of $\gamma_t/2$, we have that $M = \sum_{t=0}^{T-1} M_t \leq O(1) \cdot \sum_{t=0}^{T-1}\frac{1}{\gamma_t}\sum_{j=0}^{t-1}\gamma_j$. $\qquad\square$

**Remark.** Consider the case in which each advantage $\gamma_t$ is just $\gamma$ and we are boosting to accuracy $\epsilon$. As usual taking $T = O(\log(1/\epsilon)/\gamma^2)$ gives an error bound of $\epsilon$. With these parameters we have that $M \leq O(\log^2(1/\epsilon)/\gamma^4)$, the same asymptotic bound achieved in [LS05]. In the next section we describe a modification of the algorithm that improves this bound by essentially a factor of $\frac{1}{\gamma}$.

## 4.1 Improving efficiency by freezing extreme nodes

Here we describe a variant of the algorithm from the previous section that constructs a branching program with fewer nodes.

The algorithm requires an input parameter $\epsilon$ which is an upper bound on the desired final error of the aggregate classifier. For $t \geq 1$, after the execution of step $t-1$ of boosting, when all nodes at level $t$ have been created, each node $(\alpha, t)$ with $|\alpha| > \sqrt{\left(8\sum_{s=0}^{t-1}\gamma_s^2\right)\left(2\ln t + \ln\frac{4}{\epsilon}\right)}$ is "frozen." The algorithm commits to classifying any test examples routed to any such nodes according to $\mathrm{sgn}(\alpha)$, and these nodes are not used to generate weak hypotheses during the next round of training.

We have the following theorem about the performance of this algorithm:

**Theorem 3** *Consider running the modified booster for $T$ stages. For $t = 0, \ldots, T-1$ let the values $\gamma_1, \ldots, \gamma_T > 0$ be defined as described above, so each invocation of the weak learner on distribution $\mathcal{D}_{\beta,t}$ yields a hypothesis $h_{\beta,t}$ that has $\gamma_{\beta,t} \geq \gamma_t$. Then the final output hypothesis $h$ of the booster satisfies*

$$\mathbf{Pr}_{x\in\mathcal{D}}[h(x) \neq c(x)] \leq \frac{\epsilon}{2} + \exp\left(-\frac{1}{8}\sum_{t=0}^{T-1}\gamma_t^2\right). \tag{7}$$

*The algorithm makes $O\left(\sqrt{\left(\sum_{t=0}^{T-1}\gamma_t^2\right)\left(\ln T + \ln\frac{1}{\epsilon}\right)} \cdot \sum_{t=0}^{T-1}\frac{1}{\gamma_t}\right)$ calls to the weak learner.*

**Proof:** As in the previous proof it suffices to bound $\mathbf{Pr}_{x\in\mathcal{D}^+}[h(x) \neq 1]$. The proof of Theorem 2 gives us that if we never did any freezing, then $\mathbf{Pr}_{x\in\mathcal{D}^+}[h(x) \neq 1] \leq \exp\left(-\frac{1}{8}\sum_{t=0}^{T-1}\gamma_t^2\right)$. Now let us analyze the effect of freezing in a given stage $t < T$. Let $A_t$ be the distance from the origin past which examples are frozen in round $t$; i.e. $A_t = \sqrt{(8\sum_{s=0}^{t-1}\gamma_s^2)(2\ln t + \ln\frac{4}{\epsilon})}$. Nearly exactly the same analysis as proves (6) can be used here: for a positive example $x$ to be incorrectly frozen

in round $t$, it must be the case $X_t < -A_t$, or equivalently $Y_t < -A_t - \sum_{i=0}^{t-1} \gamma_i^2$. Thus our choice of $A_t$ gives us that $\mathbf{Pr}_{x \in \mathcal{D}^+}[x \text{ incorrectly frozen in round } t]$ is at most

$$\mathbf{Pr}[Y_t \leq -A_t - \sum_{i=0}^{t-1} \gamma_t^2] \leq \mathbf{Pr}[Y_t \leq -A_t] \leq \frac{\epsilon}{4t^2},$$

so consequently we have $\mathbf{Pr}_{x \in \mathcal{D}^+}[x \text{ ever incorrectly frozen }] \leq \frac{\epsilon}{2}$. From here we may argue as in [LS05]: we have that $\mathbf{Pr}_{x \in \mathcal{D}^+}[h(x) = 0]$ equals

$$\mathbf{Pr}_{x \in \mathcal{D}^+}[h(x = 0 \text{ and } x \text{ is frozen}] + \mathbf{Pr}_{x \in \mathcal{D}^+}[h(x) = 0 \text{ and } x \text{ is not frozen}] \leq \frac{\epsilon}{2} + \exp\left(-\frac{1}{2}\sum_{t=0}^{T-1}\gamma_t^2\right)$$

which gives (7). The bound on the number of calls to the weak learner follows from the fact that there are $O(A_t/\gamma_t)$ such calls in each stage of boosting, and the fact that $A_t \leq \sqrt{(8\sum_{s=0}^{T-1}\gamma_s^2)(2\ln T + \ln\frac{4}{\epsilon})}$ for all $t$. $\square$

It is easy to check that if $\gamma_t = \gamma$ for all $t$, taking $T = O(\log(1/\epsilon)/\gamma^2)$ the algorithm in this section will construct an $\epsilon$-accurate hypothesis that is an $O(\log^2(1/\epsilon)/\gamma^3)$-node branching program.

# 5   Extensions

## 5.1   Standard weak learners

In Sections 3 and 4, we assumed that the boosting algorithm had access to a two-sided weak learner, which is more accurate than random guessing on both the positive and the negative examples separately. To make use of a standard weak learner, which is merely more accurate than random guessing on average, we can borrow ideas from [LS05].

The idea is to force a standard weak learner to provide a hypothesis with two-sided accuracy by (a) balancing the distribution so that positive and negative examples are accorded equal importance, (b) balancing the predictions of the output of the weak learner so that it doesn't specialize on one kind of example.

**Definition 4** *Given a probability distribution $\mathcal{D}$ over examples, let $\widehat{\mathcal{D}}$ be the distribution obtained by rescaling the positive and negative examples so that they have equal weight: i.e., let $\widehat{\mathcal{D}}[S] = \frac{1}{2}\mathcal{D}^+[S] + \frac{1}{2}\mathcal{D}^-[S]$.*

**Definition 5** *Given a confidence-rated classifier $h : X \to [-1, 1]$ and a probability distribution $\mathcal{D}$ over $X$, let the balanced variant of $h$ with respect to $\mathcal{D}$ be the function $\hat{h} : X \to [-1, 1]$ defined as follows: (a) if $\mathbf{E}_{x \in \mathcal{D}}[h(x)] \geq 0$, then, for all $x \in X$, $\hat{h}(x) = \frac{h(x)+1}{\mathbf{E}_{x \in \mathcal{D}}[h(x)]+1} - 1$. (b) if $\mathbf{E}_{x \in \mathcal{D}}[h(x)] \leq 0$, then, for all $x \in X$, $\hat{h}(x) = \frac{h(x)-1}{-\mathbf{E}_{x \in \mathcal{D}}[h(x)]+1} + 1$.*

The analysis is the natural generalization of Section 5 of [LS05] to confidence-rated classifiers.

**Lemma 6** *If $\mathcal{D}$ is balanced with respect to $c$, and $h$ is a confidence-rated classifier such that $\mathbf{E}_{x \in \mathcal{D}}[h(x)c(x)] \geq \gamma$, then $\mathbf{E}_{x \in \mathcal{D}}[\hat{h}(x)c(x)] \geq \gamma/2$.*

**Proof.** Assume without loss of generality that $\mathbf{E}_{x \in \mathcal{D}}[h(x)] \geq 0$ (the other case can be handled symmetrically). By linearity of expectation

$$\mathbf{E}_{x \in \mathcal{D}}[\hat{h}(x)c(x)] = \frac{\mathbf{E}_{x \in \mathcal{D}}[h(x)c(x)]}{\mathbf{E}_{x \in \mathcal{D}}[h(x)]+1} + \mathbf{E}_{x \in \mathcal{D}}[c(x)]\left(\frac{1}{\mathbf{E}_{x \in \mathcal{D}}[h(x)]+1} - 1\right).$$

Since $\mathcal{D}$ is balanced we have $\mathbf{E}_{x \in \mathcal{D}}[c(x)] = 0$, and hence $\mathbf{E}_{x \in \mathcal{D}}[\hat{h}(x)c(x)] = \frac{\mathbf{E}_{x \in \mathcal{D}}[h(x)c(x)]}{\mathbf{E}_{x \in \mathcal{D}}[h(x)]+1}$, so the lemma follows from the fact that $\mathbf{E}_{x \in \mathcal{D}}[h(x)] \leq 1$. $\square$

We will use a standard weak learner to simulate a two-sided weak learner as follows. Given a distribution $\mathcal{D}$, the two-sided weak learner will pass $\widehat{\mathcal{D}}$ to the standard weak learner, take its output $g$, and return $h = \hat{g}$. Our next lemma analyzes this transformation.

**Lemma 7** *If* $\mathbf{E}_{x\in\widehat{\mathcal{D}}}[g(x)c(x)] \geq \gamma$, *then* $\mathbf{E}_{x\in\mathcal{D}^+}[h(x)] \geq \gamma/2$ *and* $\mathbf{E}_{x\in\mathcal{D}^-}[-h(x)] \geq \gamma/2$.

**Proof**: Lemma 6 implies that $\mathbf{E}_{x\in\widehat{\mathcal{D}}}[h(x)c(x)] \geq \gamma/2$. Expanding the definition of $\widehat{\mathcal{D}}$, we have

$$\mathbf{E}_{x\in\mathcal{D}^+}[h(x)] - \mathbf{E}_{x\in\mathcal{D}^-}[h(x)] \geq \gamma. \tag{8}$$

Since $h$ balanced $g$ with respect to $\widehat{\mathcal{D}}$ and $c$, we have $\mathbf{E}_{x\in\widehat{\mathcal{D}}}[h(x)] = 0$. Once again expanding the definition of $\widehat{\mathcal{D}}$, we get that $\mathbf{E}_{x\in\mathcal{D}^+}[h(x)] + \mathbf{E}_{x\in\mathcal{D}^-}[h(x)] = 0$ which implies $\mathbf{E}_{x\in\mathcal{D}^-}[h(x)] = -\mathbf{E}_{x\in\mathcal{D}^+}[h(x)]$ and $\mathbf{E}_{x\in\mathcal{D}^+}[h(x)] = -\mathbf{E}_{x\in\mathcal{D}^+}[h(x)]$. Substituting each of the RHS for its respective LHS in (8) completes the proof. $\square$

Lemma 7 is easily seen to imply counterparts of Theorems 2 and 3 in which the requirement of a two-sided weak learner is weakened to require only standard weak learning, but each $\gamma_t$ is replaced with $\gamma_t/2$.

## 5.2 Tolerating random classification noise

As in [LS05], noise tolerance is facilitated by the fact that the path through the network is not affected by altering the label of an example. On the other hand, balancing the distribution before passing it to the weak learner, which was needed to use a standard weak learner, may disturb the independence between the event that an example is noisy, and the random draw of $x$. This can be repaired exactly as in [KS05, LS05]; because of space constraints we omit the details.

## Footnotes

[1] The more common definition of a sub-martingale requires that $\mathbf{E}[Y_t|Y_0, ..., Y_{t-1}] \leq Y_{t-1}$, but the weaker assumption that $\mathbf{E}[Y_t|Y_{t-1}] \leq Y_{t-1}$ suffices for the concentration bounds that we need (see [ASE92, Hay05]).

## References

[AD98]   J. Aslam and S. Decatur. Specification and simulation of statistical query algorithms for efficiency and noise tolerance. *J. Comput & Syst. Sci.*, 56:191–208, 1998.

[AL88]   Dana Angluin and Philip Laird. Learning from noisy examples. *Machine Learning*, 2(4):343–370, 1988.

[ASE92]  N. Alon, J. Spencer, and P. Erdos. *The Probabilistic Method (1st ed.)*. Wiley-Interscience, New York, 1992.

[BKW03]  A. Blum, A. Kalai, and H. Wasserman. Noise-tolerant learning, the parity problem, and the statistical query model. *J. ACM*, 50(4):506–519, 2003.

[Die00]  T.G. Dietterich. An experimental comparison of three methods for constructing ensembles of decision trees: bagging, boosting, and randomization. *Machine Learning*, 40(2):139–158, 2000.

[Fre95]  Y. Freund. Boosting a weak learning algorithm by majority. *Information and Computation*, 121(2):256–285, 1995.

[FS96]   Y. Freund and R. Schapire. Experiments with a new boosting algorithm. In *ICML*, pages 148–156, 1996.

[FS97]   Y. Freund and R. E. Schapire. A decision-theoretic generalization of on-line learning and an application to boosting. *JCSS*, 55(1):119–139, 1997.

[Hay05]  T. P. Hayes. A large-deviation inequality for vector-valued martingales. 2005.

[Kea98]  M. Kearns. Efficient noise-tolerant learning from statistical queries. *JACM*, 45(6):983–1006, 1998.

[KS05]   A. Kalai and R. Servedio. Boosting in the presence of noise. *JCSS*, 71(3):266–290, 2005.

[LS05]   P. Long and R. Servedio. Martingale boosting. In *Proc. 18th Annual COLT*, pages 79–94, 2005.

[LS08]   P. Long and R. Servedio. Random classification noise defeats all convex potential boosters. In *ICML*, 2008.

[MO97]   R. Maclin and D. Opitz. An empirical evaluation of bagging and boosting. In *AAAI/IAAI*, pages 546–551, 1997.

[MR03]   R. Meir and G. Rätsch. An introduction to boosting and leveraging. In *LNAI Advanced Lectures on Machine Learning*, pages 118–183, 2003.

[RDM06]  L. Ralaivola, F. Denis, and C. Magnan. CN=CNN. In *ICML*, pages 265–272, 2006.

[Sch90]  R. Schapire. The strength of weak learnability. *Machine Learning*, 5(2):197–227, 1990.

[Sch03]  R. Schapire. *The boosting approach to machine learning: An overview*. Springer, 2003.

[SS99]   R. Schapire and Y. Singer. Improved boosting algorithms using confidence-rated predictions. *Machine Learning*, 37:297–336, 1999.
